# Inductive Regularized Learning of Kernel Functions

**Prateek Jain**
Microsoft Research Bangalore
Bangalore, India
prajain@microsoft.com

**Brian Kulis**
UC Berkeley EECS and ICSI
Berkeley, CA, USA
kulis@eecs.berkeley.edu

**Inderjit Dhillon**
UT Austin Dept. of Computer Sciences
Austin, TX, USA
inderjit@cs.utexas.edu

## Abstract

In this paper we consider the problem of semi-supervised kernel function learning. We first propose a general regularized framework for learning a kernel matrix, and then demonstrate an equivalence between our proposed kernel matrix learning framework and a general linear transformation learning problem. Our result shows that the learned kernel matrices parameterize a linear transformation kernel *function* and can be applied inductively to new data points. Furthermore, our result gives a constructive method for kernelizing most existing Mahalanobis metric learning formulations. To make our results practical for large-scale data, we modify our framework to limit the number of parameters in the optimization process. We also consider the problem of kernelized inductive dimensionality reduction in the semi-supervised setting. To this end, we introduce a novel method for this problem by considering a special case of our general kernel learning framework where we select the trace norm function as the regularizer. We empirically demonstrate that our framework learns useful kernel functions, improving the $k$-NN classification accuracy significantly in a variety of domains. Furthermore, our kernelized dimensionality reduction technique significantly reduces the dimensionality of the feature space while achieving competitive classification accuracies.

## 1    Introduction

Learning kernel functions is an ongoing research topic in machine learning that focuses on learning an appropriate kernel function for a given task. While several methods have been proposed, many of the existing techniques can only be applied transductively [1–3]; i.e., they cannot be applied inductively to new data points. Of the methods that can be applied inductively, several are either too computationally expensive for large-scale data (e.g. hyperkernels [4]) or are limited to small classes of possible learned kernels (e.g. multiple kernel learning [5]).

In this paper, we propose and analyze a general kernel matrix learning problem using provided side-information over the training data. Our learning problem regularizes the desired kernel matrix via a convex regularizer chosen from a broad class, subject to convex constraints on the kernel. While the learned kernel matrix should be able to capture the provided side-information well, it is not clear how the information can be propagated to new data points. Our first main result demonstrates that our kernel matrix learning problem is equivalent to learning a linear transformation (LT) kernel *function* (a kernel of the form $\phi(\boldsymbol{x})^T W \phi(\boldsymbol{y})$ for some matrix $W \succeq 0$) with a specific regularizer. With the appropriate representation of $W$, this result implies that the learned LT kernel function can be naturally applied to new data. Additionally, we demonstrate that a large class of Mahalanobis metric learning methods can be seen as learning an LT kernel function and so our result provides a

constructive method for kernelizing these methods. Our analysis recovers some recent kernelization results for metric learning, but also implies several new results.

As our proposed kernel learning formulation learns a kernel matrix over the training points, the memory requirements scale quadratically in the number of training points, a common issue arising in kernel methods. To alleviate such issues, we propose an additional constraint to the learning formulation to reduce the number of parameters. We prove that the equivalence to LT kernel function learning still holds with the addition of this constraint, and that the resulting formulation can be scaled to very large data sets.

We then focus on a novel application of our framework to the problem of inductive semi-supervised kernel dimensionality reduction. Our method is a special case of our kernel function learning framework with trace-norm as the regularization function. As a result, we learn *low-rank* linear transformations, which correspond to low-dimensional embeddings of high- or infinite-dimensional kernel embeddings; unlike previous kernel dimensionality methods, which are either unsupervised (kernel-PCA) or cannot easily be applied inductively to new data (spectral kernels [6]), our method intrinsically possesses both desirable properties. Furthermore, our method can handle a variety of side-information, e.g., class labels, click-through rates, etc. Finally, we validate the effectiveness of our proposed framework. We quantitatively compare several regularizers, including the trace-norm regularizer for dimensionality reduction, over standard data sets. We also apply the methods to an object recognition task in computer vision and qualitatively show results of dimensionality reduction on a handwritten digits data set.

**Related Work:** Most of the existing kernel learning methods can be classified into two broad categories. The first category includes parametric approaches, where the learned kernel function is restricted to be of a specific form and then the relevant parameters are learned according to the provided data. Prominent methods include multiple kernel learning [5], hyperkernels [4], infinite kernel learning [7], and hyper-parameter cross-validation [8]. Most of these methods either lack modeling flexibility, require non-convex optimization, or are restricted to a supervised learning scenario. The second category includes non-parametric methods, which explicitly model geometric structure in the data. Examples include spectral kernel learning [6], manifold-based kernel learning [9], and kernel target alignment [3]. However, most of these approaches are limited to the transductive setting and cannot be used to naturally generalize to new points. In comparison, our method combines both of the above approaches. We propose a general non-parametric kernel *matrix* learning framework, similar to methods of the second category. However, we show that our learned kernel matrix corresponds to a linear transformation kernel function parameterized by a PSD matrix. Hence, our method can be applied to inductive settings also without sacrificing significant modeling power. Furthermore, our methods can be applied to a variety of domains and with a variety of forms of side-information.

Existing work on learning linear transformations has largely focused on learning Mahalanobis distances; examples include [10–15], among others. POLA [13] and ITML [12] provide specialized kernelization techniques for their respective metric learning formulations. Kernelization of LMNN was discussed in [16], though it relied on a convex perturbation based formulation that can lead to suboptimal solutions. Recently, [17] showed kernelization for a class of metric learning algorithms including LMNN and NCA [15]; as we will see, our result is more general and we can prove kernelization over a larger class of problems and can also reduce the number of parameters to be learned. Independent of our work, [18] recently proved a representer type of theorem for spectral regularization functions. However, the framework they consider is different than ours in that they are interested in sensing the underlying high-dimensional matrix using given measurements.

Kernel dimensionality reduction methods can generally be divided into two categories: 1) semi-supervised dimensionality reduction in the transductive setting, 2) supervised dimensionality reduction in the inductive setting. Methods in the first category include the incomplete Cholesky decomposition [19], colored maximum variance unfolding [20], manifold preserving semi-supervised dimensionality reduction [21]. Methods in the second category include the kernel dimensionality reduction method [22] and Gaussian Process latent variable models [23]. Kernel PCA [24] reduces the dimensionality in the inductive unsupervised setting, while various manifold learning methods can reduce the dimensionality but only in the unsupervised transductive setting. In contrast, our dimensionality reduction method, which is an instantiation of our general kernel learning framework, can perform kernel dimensionality reduction simultaneously in both the semi-supervised as well as the inductive setting. Additionally, it can capture the manifold structure using an appropriate baseline kernel function such as the one proposed by [25].

## 2 Learning Framework

Given an input kernel function $\kappa : \mathbb{R}^d \times \mathbb{R}^d \to \mathbb{R}$, and some side-information over a set of points $X = \{\boldsymbol{x}_1, \boldsymbol{x}_2, \ldots, \boldsymbol{x}_n\}$ the goal is to learn a new kernel function $\kappa_W$ that is regularized against $\kappa$ but incorporates the provided side-information (the use of the subscript $W$ will become clear later). The initial kernel function $\kappa$ is of the form $\kappa(\boldsymbol{x}, \boldsymbol{y}) = \phi(\boldsymbol{x})^T \phi(\boldsymbol{y})$ for some mapping $\phi$. Throughout the rest of this paper, we will denote $\phi_i$ as shorthand for $\phi(\boldsymbol{x}_i)$, i.e., data point $\boldsymbol{x}_i$ after applying the mapping $\phi$. We will also assume that the data vectors in $X$ have been mapped via $\phi$, resulting in $\Phi = \{\phi_1, \phi_2, \ldots, \phi_n\}$. Learning a kernel function from the provided side-information is an ill-posed problem since infinitely many such kernels can satisfy the provided supervision. A common approach is to formulate a transductive learning problem to learn a new kernel matrix over the training data. Denoting the input kernel matrix $K$ as $K = \Phi^T \Phi$, we aim to learn a new kernel matrix $K_W$ that is regularized against $K$ while satisfying the available side-information. In this work, we study the following optimization problem:

$$\min_{K_W \succeq 0} \quad f(K^{-1/2} K_W K^{-1/2}) \qquad \text{s.t.} \quad g_i(K_W) \leq b_i, \; 1 \leq i \leq m, \tag{1}$$

where $f$ and $g_i$ are functions from $\mathbb{R}^{n \times n} \to \mathbb{R}$. We call $f$ the *regularizer* and the $g_i$ the *constraints*. Note that if $f$ and constraints $g_i$'s are all convex functions, then the above problem can be solved optimally using standard convex optimization algorithms. Note that our results will also hold for unconstrained variants of the above problem, as well as variants that incorporate slack variables.

In general, such learning formulations are limited in that the learned kernel cannot readily be applied to new data points. However, we will show that the above proposed problem is equivalent to learning linear transformation (LT) kernel functions. Formally, an LT kernel function $\kappa_W$ is a kernel function of the form $\kappa_W(\boldsymbol{x}, \boldsymbol{y}) = \phi(\boldsymbol{x})^T W \phi(\boldsymbol{y})$, where $W$ is a positive semi-definite (PSD) matrix; we can think of the LT kernel as describing the linear transformation $\phi_i \to W^{1/2} \phi_i$. A natural way to learn an LT kernel function would be to learn the parameterization matrix $W$ using the provided side-information. To this end, we consider the following problem:

$$\min_{W \succeq 0} \quad f(W) \qquad \text{s.t.} \quad g_i(\Phi^T W \Phi) \leq b_i, \; 1 \leq i \leq m, \tag{2}$$

where, as before, the function $f$ is the regularizer and the functions $g_i$ are the constraints that encode the side information. The constraints $g_i$ are assumed to be a function of the matrix $\Phi^T W \Phi$ of learned kernel values over the training data. We make two observations about this problem: first, for data mapped to high-dimensional spaces via kernel functions, this problem is seemingly impossible to optimize since the size of $W$ grows quadratically with the dimensionality. We will show that (2) need not explicitly be solved for learning an LT kernel function. Second, most Mahalanobis metric learning methods may be viewed as a special case of the above framework, and we will discuss some of them throughout the paper.

### 2.1 Examples of Regularizers and Constraints

To make the kernel learning optimization problem concrete, we discuss a few examples of possible regularizers and constraints.

For the regularizer $f(A) = \frac{1}{2} \|A - I\|_F^2$, the resulting kernel learning objective can be equivalently expressed as minimizing $\frac{1}{2} \|K^{-1} K_W - I\|_F^2$. Thus, the goal is to keep the learned kernel close to the input kernel subject to the constraints in $g_i$. Similarly, for $f(A) = \text{tr}(A - I)$, the resulting objective can be expressed as minimizing $\text{tr}(K^{-1} K_W - I)$. Another interesting regularizer is $f(A) = \text{tr}(A) - \log \det(A)$. In this case, the resulting objective is to minimize the LogDet divergence $D_{\ell d}(K_W, K)$ subject to the constraints given by $g_i$. For linear $g_i$, this problem was studied in [12, 26].

In terms of constraints, pairwise squared Euclidean distance constraint between a pair of points $(\phi_i, \phi_j)$ in feature space can be formulated as $K_W(i, i) + K_W(j, j) - 2K_W(i, j) \geq b$ or $K_W(i, i) + K_W(j, j) - 2K_W(i, j) \leq b$; this constraint is clearly linear in the entries of $K_W$. Similarity constraints can be represented as $K_W(i, j) \leq b$ or $K_W(i, j) \geq b$ and are also linear in $K_W$. Relative distance constraints over a triplet $(\phi_i, \phi_j, \phi_k)$ specify that $\phi_i$ should be closer to $\phi_j$ than $\phi_k$, and are often used in metric learning formulations and ranking problems; such constraints can be easily formulated within our framework. Finally, non-parametric probability estimation constraints can be used to constrain the conditional probability of a class $c$ given a data point $\phi_i$,

$$\pm p(c|x) = \pm \frac{\sum_{j \in c} K_W(i, j)}{\sum_{t=1}^C \sum_{j \in t} K_W(i, j)} \geq b,$$

where $C$ is the number of classes. This constraint can be written as a linear constraint over $K_W$ after appropriate manipulation.

## 3 Analysis

We are now ready to analyze the connection between problems (1) and (2). We will show that the solutions to the two problems are equivalent, in the sense that by optimally solving one of the problems, the solution to the other can be computed in closed form. More importantly, this result will yield insight into the type of kernel that is learned by the kernel learning problem.

We begin by defining the class of regularizers considered in our analysis. Note that each of the example regularizers discussed earlier satisfy the following definition of spectral functions.

**Definition 3.1.** *We say that $f : \mathbb{R}^{n \times n} \to \mathbb{R}$ is a **spectral function** if $f(A) = \sum_i f_s(\lambda_i)$, where $\lambda_1, ..., \lambda_n$ are the eigenvalues of $A$ and $f_s : \mathbb{R} \to \mathbb{R}$ is a real-valued function over the reals. Note that if $f_s$ is a convex function over the reals, then $f$ is also convex.*

### 3.1 Learning Linear Transformation Kernels

Now we present our main result, i.e., for a spectral function $f$, problems (1) and (2) are equivalent.

**Theorem 1.** *Let $K \succ 0$ be an invertible matrix, $f$ be a spectral function and denote the global minima of the corresponding scalar function $f_s$ as $\alpha$. Let $W^*$ be an optimal solution to (2) and $K_W^*$ be an optimal solution to (1). Then,*

$$W^* = \alpha I + \Phi S^* \Phi^T,$$

*where $S^* = K^{-1}(K_W^* - \alpha K)K^{-1}$. Furthermore, $K_W^* = \Phi^T W^* \Phi$.*

The first part of the theorem demonstrates that, given an optimal solution $K_W^*$ to (1), one can construct the corresponding solution $W^*$ to (2), while the second part shows the reverse (this also demonstrates why $W$ is used in the subscript of the learned kernel). The proof of this theorem appears in the supplementary material. The main idea behind the proof is to first show that the optimal solution to (2) is always of the form $W = \alpha I + \Phi S \Phi^T$, and then we obtain the closed form expression for $S$ using algebraic manipulations.

As a first consequence of this result, we can achieve induction over the learned kernels. Given that $K_W = \Phi^T W \Phi$, we can see that the learned kernel function is a linear transformation kernel; that is, $\kappa_W(\phi_i, \phi_j) = \phi_i^T W \phi_j$. Given a pairs of new data points $\phi_{n_1}$ and $\phi_{n_2}$, we use the fact that the learned kernel is a linear transformation kernel, along with the first result of the theorem $(W = \alpha I + \Phi S \Phi^T)$ to compute the learned kernel as:

$$\kappa_W(\boldsymbol{x}_{n_1}, \boldsymbol{x}_{n_2}) = \phi_{n_1}^T W \phi_{n_2} = \alpha \kappa(\boldsymbol{x}_{n_1}, \boldsymbol{x}_{n_2}) + \sum_{i,j=1}^n S_{ij} \kappa(\boldsymbol{x}_{n_1}, \boldsymbol{x}_i) \kappa(\boldsymbol{x}_j, \boldsymbol{x}_{n_2}). \tag{3}$$

As mentioned in Section 2, many Mahalanobis metric learning methods can be viewed as a special case of (2). Therefore, a corollary of Theorem 1 is that we can constructively apply these metric learning methods in kernel space by solving their corresponding kernel learning problem, and then compute the learned metrics via (3). Thus, $W$ need not explicitly be constructed to learn the LT kernel. Kernelization of Mahalanobis metric learning has previously been established for some special cases; our results generalize and extend previous methods, as well as provide simpler techniques in some cases. Below, we elaborate with some special cases.

**Example 1 [Information Theoretic Metric Learning (ITML)]:** [12] proposed the following Mahalanobis metric learning problem formulation:

$$\min_{W \succeq 0} \mathrm{Tr}(W) - \log \det(W), \quad s.t. \quad d_W(\phi_i, \phi_j) \le b_{ij}, \ (i,j) \in \mathcal{S}, \quad d_W(\phi_i, \phi_j) \ge b_{ij}, \ (i,j) \in \mathcal{D},$$

where $\mathcal{S}$ and $\mathcal{D}$ specify pairs of similar and dissimilar points, respectively, and $d_W(\phi_i, \phi_j) = (\phi_i - \phi_j)^T W (\phi_i - \phi_j)$ is the *Mahalanobis distance* between $\phi_i$ and $\phi_j$. ITML is an instantiation of our framework with regularizer $f(A) = \mathrm{tr}(A) - \log \det(A)$ and pairwise distance constraints encoded as the $g_i$ functions. Furthermore, it is straightforward to show that $f$ is a convex spectral function with global optima $\alpha = 1$, so the optimal $W$ can be learned implicitly using (1). The corresponding kernel learning optimization problem simplifies to:

$$\min_{K_W} \quad D_{\ell d}(K_W, K) \quad \text{s.t.} \quad g_i(K_W) \le b_i, \ 1 \le i \le m, \tag{4}$$

where $D_{\ell d}(K_W, K) = \text{tr}(K_W K^{-1}) - \log \det(K_W K^{-1}) - n$ is the LogDet divergence [12], and the positive definiteness of $K_W$ is satisfied automatically. This recovers the kernelized metric learning problem analyzed in [12], where kernelization for this special case was established and an iterative projection algorithm for optimization was developed. Note that, in the analysis of [12], the $g_i$ were limited to similarity and dissimilarity constraints; our result is therefore more general than the existing kernelization result, even for this special case.

**Example 2 [Pseudo Online Metric Learning (POLA)]:** [13] proposed the following metric learning formulation:

$$\min_{W \succeq 0} \|W\|_F^2, \quad \text{s.t. } y_{ij}(b - d_W(\phi_i, \phi_j)) \geq 1, \ \forall (i,j) \in \mathcal{P},$$

where $y_{ij} = 1$ if $\phi_i$ and $\phi_j$ are similar, and $y_{ij} = -1$ if $\phi_i$ and $\phi_j$ are dissimilar. $\mathcal{P}$ is a set of pairs of points with known distance constraints. POLA is an instantiation of (2) with $f(A) = \frac{1}{2}\|A\|_F^2$ and side-information available in the form of pair-wise distance constraints. Note that the regularizer $f(A) = \frac{1}{2}\|A\|^2$ was also employed in [2, 27], and these methods also fall under our general formulation. In this case, $f$ is once again a convex spectral function, and its global minima is $\alpha = 0$, so we can use (1) to solve for the learned kernel $K_W$ as

$$\min_{K_W} \ \|K_W K^{-1}\|_F^2 \quad \text{s.t.} \quad g_i(K_W) \leq b_i, \ 1 \leq i \leq m, \quad K_W \succeq 0. \quad (5)$$

The constraints $g_i$ for this problem can be easily constructed by re-writing each of POLA's constraints as a function of $\Phi^T W \Phi$. Note that the above approach for kernelization is much simpler than the method suggested in [13], which involves a kernelized Gram-Schmidt procedure at each step of the algorithm.

**Other Examples:** The above two examples show that our analysis recovers two well-known kernelization results for Mahalanobis metric learning. However, there are several other metric learning approaches that fall into our framework as well, including the large margin nearest neighbor metric learning method (LMNN) [11] and maximally collapsing metric learning (MCML) [14], both of which can be seen as instantiations of our learning framework with a constant $f$, as well as relevant component analysis (RCA) [28] and Xing et al.'s Mahalanobis metric learning method for clustering [10]. Given lack of space, we cannot detail the kernelization of all these methods, but they follow in the same manner as in the above two examples. In particular, each of these methods may be run in kernel space, and our analysis yields new insights into these methods; for example, kernelization of LMNN [11] using Theorem 1 avoids the convex perturbation analysis in [16] that leads to suboptimal solutions in some cases.

## 3.2 Parameter Reduction

One of the drawbacks to Theorem 1 is that the size of the matrices $K_W$ and $S$ are $n \times n$, and thus grow quadratically with the number of data points. We would like to have a way to restrict our optimization over a smaller number of parameters, so we now discuss a generalization of (2) by introducing an additional constraint to make it possible to reduce the number of parameters to learn, permitting scalability to data sets with many training points *and* with very high dimensionality.

Theorem 1 shows that the optimal $K_W^*$ is of the form $\Phi^T W^* \Phi = \alpha K + KS^* K$. In order to accommodate fewer parameters to learn, a natural option is to replace the unknown $S$ matrix with a *low-rank* matrix $JLJ^T$, where $J \in \mathbb{R}^{n \times r}$ is a pre-specified matrix, $L \in \mathbb{R}^{r \times r}$ is unknown (we use $L$ instead of $S$ to emphasize that $S$ is of size $n \times n$ whereas $L$ is $r \times r$), and the rank $r$ is a parameter of the algorithm. Then, we will explicitly enforce that the learned kernel is of this form.

By plugging in $K_W = \alpha K + KSK$ into (1) and replacing $S$ with $JLJ^T$, the resulting optimization problem is given by:

$$\min_{L \succeq 0} f(\alpha I + K^{1/2}JLJ^T K^{1/2}) \quad \text{s.t. } g_i(\alpha K + KJLJ^T K) \leq b_i, \ 1 \leq i \leq m. \quad (6)$$

While the above problem involves just $r \times r$ variables, the functions $f$ and $g_i$'s are applied to $n \times n$ matrices and therefore the problem may still be computationally expensive to optimize. Below, we show that for any spectral function $f$ and linear constraints $g_i(K_W) = \text{Tr}(C_i K_W)$, (6) reduces to a problem that applies $f$ and $g_i$'s to $r \times r$ matrices only, which provides significant scalability.

**Theorem 2.** *Let $K = \Phi^T \Phi \succ 0$ and $J \in \mathbb{R}^{n \times r}$. Also, let the regularization function $f$ be a spectral function (see Definition 3.1) such that the corresponding scalar function $f_s$ has a global minima at $\alpha$. Then problem* (6) *is equivalent to the following problem:*

$$\min_{L \succeq -\alpha(K^J)^{-1}} f((K^J)^{-1/2}(\alpha K^J + K^J L K^J)(K^J)^{-1/2}),$$
$$\text{s.t. } Tr(LJ^T KC_i KJ) \leq b_i - Tr(\alpha KC_i), \ 1 \leq i \leq m. \quad (7)$$

Note that (7) is over $r \times r$ matrices (after initial pre-processing) and is in fact similar to the kernel learning problem (1), but with a kernel $K^J$ of smaller size $r \times r$, $r \ll n$. A proof of the above theorem is in the supplementary material, and follows by showing that for spectral functions the objective functions of the two problems can be shown to differ by a universal constant.

Similar to (1), we can show that (6) is also equivalent to linear transformation kernel function learning. This enables us to naturally apply the above kernel learning problem in the inductive setting. We provide a proof of the following theorem in the supplementary material.

**Theorem 3.** *Consider* (6) *with a spectral function $f$ so that corresponding scalar function $f_s$ has a global minima at $\alpha$ and let $K \succ 0$ be invertible. Then,* (6) *and* (7) *are equivalent to the following linear transformation kernel learning problem (analogous to the connection between* (1) *and* (2)*):*

$$\min_{W \succeq 0, L} \quad f(W) \quad s.t. \quad Tr(\Phi^T W \Phi) \leq b_i, \ 1 \leq i \leq m, \quad W = \alpha I + XJLJX^T. \quad (8)$$

Note that, in contrast to (2), where the last constraint over $W$ is achieved automatically, (8) requires that constraint should be satisfied during the optimization process which leads to a reduced number of parameters for our kernel learning problem. The above theorem shows that our reduced parameters kernel learning method (6) also implicitly learns a linear transformation kernel function, hence we can generalize the learned kernel to unseen data points using an expression similar to (3).

The parameter reduction approach presented in this section depends critically on the choice of $J$. A few simple heuristics for choosing $J$ beyond choosing a subset of the points from $\Phi$ include a randomly sampled coefficient matrix or clustering $\Phi$ into $r$ clusters such that $J$ is the cluster membership indicator function. Also note that using this parameter reduction technique, we can scale the optimization to kernel learning problems with millions of points of more. For example, we have applied a special case of this scalable framework to learn kernels over data sets containing nearly half a million images, as well as the MNIST data set of 60,000 data points [29].

## 4 Trace-norm based Inductive Semi-supervised Kernel Dimensionality Reduction (Trace-SSIKDR)

We now consider applying our framework to the scenario of semi-supervised kernel dimensionality reduction, which provides a novel and practical application of our framework. While there exists a variety of methods for kernel dimensionality reduction, most of these methods are unsupervised (e.g. kernel-PCA) or are restricted to the transductive setting. In contrast, we can use our kernel learning framework to learn a low-rank transformation of the feature vectors implicitly that in turn provides a low-dimensional embedding of the dataset. Furthermore, our framework permits a variety of side-information such as pair-wise or relative distance constraints, beyond the class label information allowed by existing transductive methods.

We describe our method starting from the linear transformation problem. Our goal is to learn a low-rank linear transformation $W$ whose corresponding low-dimensional mapped embedding of $\phi_i$ is $W^{1/2}\phi_i$. Even when the dimensionality of $\phi_i$ is very large, if the rank of $W$ is low enough, then the mapped embedding will have small dimensionality. With that in mind, a possible regularizer could be the rank, i.e., $f(A) = \text{rank}(A)$; one can easily show that this satisfies the definition of a spectral function. Unfortunately, optimization is intractable in general with the non-convex rank function, so we use the trace-norm relaxation for the matrix rank function, i.e., we set $f(A) = \text{Tr}(A)$. This function has been extensively studied as a relaxation for the rank function [30], and it satisfies the definition of a spectral function (with $\alpha = 0$). We also add a small Frobenius norm regularization for ease of optimization (this does not affect the spectral property of the regularization function). Then using Theorem 1, the resulting relaxed kernel learning problem is:

$$\min_{K_W \succeq 0} \tau \text{Tr}(K^{-1/2} K_W K^{-1/2}) + \|K^{-1/2} K_W K^{-1/2}\|_F^2 \quad s.t. \ \text{Tr}(C_i K_W) \leq b_i, \ 1 \leq i \leq m, \quad (9)$$

where $\tau > 0$ is a parameter. The above problem can be solved using a method based on Uzawa's inexact algorithm, similar to [31].

We briefly describe the steps taken by our method at each iteration. For simplicity, denote $\tilde{K} = K^{-1/2} K_W K^{-1/2}$; we will optimize with respect to $\tilde{K}$ instead of $K_W$. Let $\tilde{K}^t$ be the $t$-th iterate. Associate variable $z_i^t, 1 \leq i \leq m$ with each constraint at each iteration $t$, and let $z_i^0 = 0, \forall i$. Let $\delta_t$

Table 1: UCI Datasets: accuracy achieved by various methods. The numbers in parentheses show the rank of the corresponding learned kernels. Trace-SSIKDR achieves accuracy comparable to Frob (Frobenius norm regularization) and ITML (LogDet regularization) with a significantly smaller rank.

| Dataset\Method | Gaussian | Frob | ITML | Frob LR | ITML LR-pre | ITML LR-post | Trace-SSIKDR |
|---|---|---|---|---|---|---|---|
| Iris | 0.99(40) | 0.99(27) | 0.99(40) | 0.91(4) | 0.93(4) | 0.99(4) | 0.99(4) |
| Wine | 0.80(105) | 0.94(36) | 0.99(105) | 0.72(11) | 0.85(11) | 0.46(11) | 0.94(11) |
| Ionosphere | 0.94(337) | 0.98(64) | 0.98(337) | 0.98(19) | 0.98(19) | 0.93(19) | 0.99(19) |
| Soybean | 0.89(624) | 0.96(96) | 0.96(624) | 0.44(40) | 0.87(40) | 0.35(40) | 0.96(40) |
| Diabetes | 0.75(251) | 0.74(154) | 0.76(251) | 0.67(14) | 0.62(14) | 0.73(14) | 0.74(14) |
| Balance-scale | 0.93(156) | 0.96(106) | 0.97(156) | 0.97(10) | 0.80(10) | 0.82(10) | 0.97(10) |
| Breast-cancer | 0.72(259) | 0.73(61) | 0.78(259) | 0.69(21) | 0.68(21) | 0.68(21) | 0.75(21) |
| Spectf-heart | 0.74(267) | 0.87(39) | 0.84(267) | 0.84(22) | 0.89(22) | 0.89(22) | 0.84(22) |
| Heart-c | 0.68(228) | 0.78(62) | 0.79(228) | 0.73(39) | 0.61(39) | 0.55(39) | 0.78(39) |
| Heart-h | 0.59(117) | 0.69(71) | 0.70(117) | 0.56(31) | 0.30(31) | 0.56(31) | 0.68(31) |

be the step size at iteration $t$. The algorithm performs the following updates:

$$U\Sigma U^T \leftarrow K^{1/2}\bigg(\sum_i z_i^{t-1} C_i\bigg)K^{1/2}, \qquad \tilde{K}^t \leftarrow U\max(\Sigma - \tau I, 0)U^T,$$

$$z_i^t \leftarrow z_i^{t-1} - \delta \max(\text{Tr}(C_i K^{1/2}\tilde{K}^t K^{1/2}) - b_i, 0), \forall i.$$

The above updates require computation of $K^{1/2}$ which is expensive for large high-rank matrices. However, using elementary linear algebra we can show that $\tilde{K}$ and the learned kernel function can be computed efficiently without computing $K^{1/2}$ by maintaining $S = K^{-1/2}\tilde{K}K^{-1/2}$ from step to step. Algorithm 1 details an efficient method for optimizing (9) and returns matrices $\Sigma_k$, $D_k$ and $V_k$ all of which are contain only $O(nk)$ parameters, where $k$ is the rank of $\tilde{K}^t$, which changes from iteration to iteration. Note that step 4 of the algorithm computes $k$ singular vectors and requires $O(nk^2)$. Since $k$ is typically significantly smaller than $n$, the computational cost will be significantly smaller than computing the whole SVD. Note that the learned embedding $\phi_i \rightarrow \tilde{K}^{1/2}K^{-1/2}\boldsymbol{k}_i$, where $\boldsymbol{k}_i$ is a vector of input kernel function values between $\phi_i$ and the training data, can be computed efficiently as $\phi_i \rightarrow \Sigma_k^{1/2}D_k V_k \boldsymbol{k}_i$, which does not require $K^{1/2}$ explicitly. We defer the proof of correctness for Algorithm 1 to the supplementary material.

---

**Algorithm 1** Trace-SSIKDR

---

**Require:** $K, (C_i, b_i), 1 \le i \le m, \tau, \delta$
 1: **Initialize:** $z_i^0 = 0$, $t = 0$
 2: **repeat**
 3:     $t = t + 1$
 4:     Compute $V_k$ and $\Sigma_k$, the top $k$ eigenvectors and eigenvalues of $\left(\sum_i z_i^{t-1} C_i\right)K$, where $k = \text{argmax}_j \sigma_j > \tau$
 5:     $D_k(i,i) \leftarrow 1/\boldsymbol{v}_i^T K \boldsymbol{v}_i, 1 \le i \le k$
 6:     $z_i^t \leftarrow z_i^{t-1} - \delta \max(\text{Tr}(C_i K V_k D_k \Sigma_k D_k V_k^T K) - b_i, 0), \forall i.$     $//S^t = V_k D_k \Sigma_k D_k V_k^T$
 7: **until** Convergence
 8: **Return** $\Sigma_k, D_k, V_k$

---

## 5 Experimental Results

We now present empirical evaluation of our kernel learning framework and our semi-supervised kernel dimensionality approach when applied in conjunction with $k$-nearest neighbor classification. In particular, using different regularization functions, we show that our framework can be used to obtain significantly better kernels than the baseline kernels for $k$-NN classification. Additionally, we show that our semi-supervised kernel dimensionality reduction approach achieves comparable accuracy while significantly reducing the dimensionality of the linear mapping.

**UCI Datasets:** First, we evaluate the performance of our kernel learning framework on standard UCI datasets. We measure accuracy of the learned kernels using 5-NN classification with two-fold cross validation averaged over 10 runs. For training, we use pairwise (dis)similarity constraints as described in Section 2.1. We select parameters $l$ and $u$ (right-hand side of the pairwise constraints) using $5^{th}$ and $95^{th}$ percentiles of all the pairwise distances between points from the training dataset.

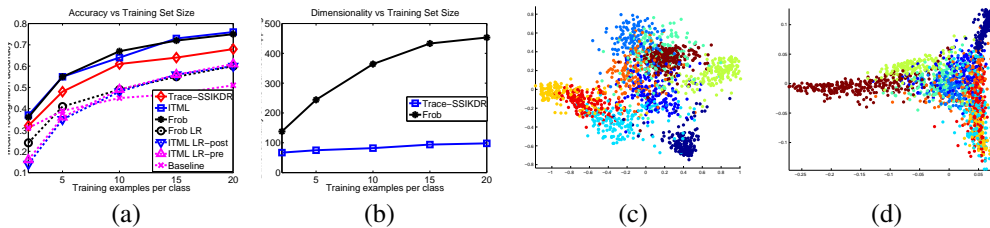

|  | (a) | (b) | (c) | (d) |

Figure 1: **(a)**: Mean classification accuracy on Caltech101 dataset obtained by 1-NN classification with learned kernels obtained by various methods. **(b)**: Rank of the learned kernel functions obtained by various methods. The rank of the learned kernel function is same as the reduced dimensionality of the dataset. **(c)**: Two-dimensional embedding of 2000 USPS digits obtained using our method Trace-SSIKDR for a training set of just 100 USPS digits. Note that we use the **inductive** setting here and the embedding is color coded according to the underlying digit. **(d)**: Embedding of the USPS digits dataset obtained using kernel-PCA.

Table 4 shows the 5-NN classification accuracies achieved by our kernel learning framework with different regularization functions. *Gaussian* represents the baseline Gaussian kernel, *Frob* represents an instantiation of our framework with Frobenius norm ($f(A) = \|A\|_F^2$) regularization, while *ITML* corresponds to the LogDet regularization ($f(A) = \mathrm{Tr}(A) - \log \det(A)$ ). For the latter case, our formulation is same as formulation proposed by [12]. Note that for almost all the datasets (except Iris and Diabetes), both Frob and ITML improve upon the baseline Gaussian kernel significantly.

We also compare our semi-supervised dimensionality reduction method *Trace-SSIKDR* (see Section 4) with baseline kernel dimensionality reduction methods *Frob LR*, *ITML LR-pre*, and *ITML LR-post*. Frob LR reduces the rank of the learned matrix $W$ (equivalently, it reduces the dimensionality) using Frobenius norm regularization by taking the top eigenvectors. Similarly, ITML LR-post reduces the rank of the learned kernel matrix obtained using ITML by taking its top eigenvectors. ITML LR-pre reduces the rank of the kernel function by reducing the rank of the *training* kernel matrix. The learned linear transformation $W$ (or equivalently, the learned kernel function) should have the same rank as that of *training* kernel matrix as the LogDet divergence preserves the range space of the input kernel. We fix the rank of the learned $W$ for Frob LR, ITML LR-pre, ITML LR-post as the rank of the transformation $W$ obtained by our Trace-SSIKDR method. Note that Trace-SSIKDR achieves accuracies similar to Frob and ITML, while decreasing the rank significantly. Furthermore, it is significantly better than the corresponding baseline dimensionality reduction methods.

**Caltech-101:** Next, we evaluate our kernel learning framework on the Caltech-101 dataset, a benchmark object recognition dataset containing over 3000 images. Here, we compare various methods using 1-NN classification method and the accuracy is measured in terms of the mean recognition accuracy per class. We use a pool of 30 images per class for our experiments, out of which a varying number of random images are selected for training and the remaining are used for testing the learned kernel function. The baseline kernel function is selected to be the sum of four different kernel functions: PMK [32], SPMK [33], Geoblur-1 and Geoblur-2 [34]. Figure 1 (a) shows the accuracy achieved by various methods (acronyms represent the same methods as described in the previous section). Clearly, ITML and Frob (which are specific instances of our framework) are able to learn significantly more accurate kernel functions than the baseline kernel function. Furthermore, our Trace-SSIKDR method is able to achieve reasonable accuracy while reducing the rank of the kernel function significantly (Figure 1 (b)). Also note that Trace-SSIKDR achieves significantly better accuracy than Frob LR, ITML LR-pre and ITML LR-post, although all of these methods have the same rank as Trace-SSIKDR.

**USPS Digits:** Finally, we qualitatively evaluate our dimensionality reduction method on the USPS digits dataset. Here, we train our method using 100 examples to learn a linear mapping to two dimensions, i.e., a rank-2 matrix $W$. For the baseline kernel, we use the data-dependent kernel function proposed by [25] that also takes data's manifold structure into account. We then embed 2000 (unseen) test examples into two dimensions using our learned low-rank transformation. Figure 1 (c) shows the embedding obtained by our Trace-SSIKDR method, while Figure 1 (d) shows the embedding obtained by the kernel-PCA algorithm. Each point is color coded according to the underlying digit. Note that our method is able to separate out most of the digits even in 2D, and is significantly better than the embedding obtained using kernel-PCA.

**Acknowledgements:** This research was supported in part by NSF grant CCF-0728879.

# References

[1] K. Tsuda, G. Rätsch, and M. K. Warmuth. Matrix exponentiated gradient updates for on-line learning and Bregman projection. *JMLR*, 6:995–1018, 2005.

[2] J. T. Kwok and I. W. Tsang. Learning with idealized kernels. In *ICML*, 2003.

[3] N. Cristianini, J. Shawe-Taylor, A. Elisseeff, and J. Kandola. On kernel-target alignment. In *NIPS*, 2001.

[4] C. S. Ong, A. J. Smola, and R. C. Williamson. Learning the kernel with hyperkernels. *JMLR*, 6:1043–1071, 2005.

[5] G. R. G. Lanckriet, N. Cristianini, P. L. Bartlett, L. El Ghaoui, and M. I. Jordan. Learning the kernel matrix with semidefinite programming. *JMLR*, 5:27–72, 2004.

[6] Xiaojin Zhu, Jaz Kandola, Zoubin Ghahramani, and John Lafferty. Nonparametric transforms of graph kernels for semi-supervised learning. In Lawrence K. Saul, Yair Weiss, and Léon Bottou, editors, *NIPS*, volume 17, pages 1641–1648, 2005.

[7] Peter V. Gehler and Sebastian Nowozin. Let the kernel figure it out; principled learning of pre-processing for kernel classifiers. In *CVPR*, pages 2836–2843, 2009.

[8] Matthias Seeger. Cross-validation optimization for large scale hierarchical classification kernel methods. In *NIPS*, pages 1233–1240, 2006.

[9] Yoshua Bengio, Olivier Delalleau, Nicolas Le Roux, Jean-Francois Paiement, Pascal Vincent, and Marie Ouimet. Learning eigenfunctions links spectral embedding and kernel PCA. *Neural Computation*, 16(10):2197–2219, 2004.

[10] E. P. Xing, A. Y. Ng, M. I. Jordan, and S. J. Russell. Distance metric learning with application to clustering with side-information. In *NIPS*, pages 505–512, 2002.

[11] K. Q. Weinberger, J. Blitzer, and L. K. Saul. Distance metric learning for large margin nearest neighbor classification. In *NIPS*, 2005.

[12] J. V. Davis, B. Kulis, P. Jain, S. Sra, and I. S. Dhillon. Information-theoretic metric learning. In *ICML*, pages 209–216, 2007.

[13] S. Shalev-Shwartz, Y. Singer, and A. Y. Ng. Online and batch learning of pseudo-metrics. In *ICML*, 2004.

[14] A. Globerson and S. T. Roweis. Metric learning by collapsing classes. In *NIPS*, 2005.

[15] J. Goldberger, S. Roweis, G. Hinton, and R. Salakhutdinov. Neighbourhood component analysis. In *NIPS*, 2004.

[16] B. Kulis, S. Sra, and I. S. Dhillon. Convex perturbations for scalable semidefinite programming. In *AISTATS*, 2009.

[17] R. Chatpatanasiri, T. Korsrilabutr, P. Tangchanachaianan, and B. Kijsirikul. On kernelization of supervised Mahalanobis distance learners, 2008.

[18] Andreas Argyriou, Charles A. Micchelli, and Massimiliano Pontil. On spectral learning. *JMLR*, 11:935–953, 2010.

[19] F. R. Bach and M. I. Jordan. Predictive low-rank decomposition for kernel methods. In *ICML*, pages 33–40, 2005.

[20] L. Song, A. Smola, K. M. Borgwardt, and A. Gretton. Colored maximum variance unfolding. In *NIPS*, pages 1385–1392, 2007.

[21] Y. Song, F. Nie, C. Zhang, and S. Xiang. A unified framework for semi-supervised dimensionality reduction. *Pattern Recognition*, 41(9):2789–2799, 2008.

[22] K. Fukumizu, F. R. Bach, and M. I. Jordan. Kernel dimensionality reduction for supervised learning. In *NIPS*, 2003.

[23] R. Urtasun and T. Darrell. Discriminative gaussian process latent variable model for classification. In *ICML*, pages 927–934, 2007.

[24] S. Mika, B. Schölkopf, A. J. Smola, K. Müller, M. Scholz, and G. Rätsch. Kernel pca and de-noising in feature spaces. In *NIPS*, pages 536–542, 1998.

[25] V. Sindhwani, P. Niyogi, and M. Belkin. Beyond the point cloud: from transductive to semi-supervised learning. In *ICML*, pages 824–831, 2005.

[26] Brian Kulis, Mátyás Sustik, and Inderjit S. Dhillon. Learning low-rank kernel matrices. In *ICML*, pages 505–512, 2006.

[27] Matthew Schultz and Thorsten Joachims. Learning a distance metric from relative comparisons. In *NIPS*, 2003.

[28] A. Bar-Hillel, T. Hertz, N. Shental, and D. Weinshall. Learning a mahalanobis metric from equivalence constraints. *JMLR*, 6:937–965, 2005.

[29] P. Jain, B. Kulis, and K. Grauman. Fast image search for learned metrics. In *CVPR*, 2008.

[30] B. Recht, M. Fazel, and P. A. Parrilo. Guaranteed minimum-rank solutions of linear matrix equations via nuclear norm minimization, 2007.

[31] J. Cai, E. J. Candes, and Z. Shen. A singular value thresholding algorithm for matrix completion, 2008.

[32] K. Grauman and T. Darrell. The Pyramid Match Kernel: Efficient learning with sets of features. *Journal of Machine Learning Research (JMLR)*, 8:725–760, April 2007.

[33] S. Lazebnik, C. Schmid, and J. Ponce. Beyond bags of features: Spatial pyramid matching for recognizing natural scene categories. In *CVPR*, pages 2169–2178, 2006.

[34] A. C. Berg and J. Malik. Geometric blur for template matching. In *CVPR*, pages 607–614, 2001.

